# A Model for Learning Variance Components of Natural Images

**Yan Karklin**
`yan+@cs.cmu.edu`

**Michael S. Lewicki**[*]
`lewicki@cnbc.cmu.edu`

Computer Science Department &
Center for the Neural Basis of Cognition
Carnegie Mellon University

## Abstract

We present a hierarchical Bayesian model for learning efficient codes of higher-order structure in natural images. The model, a non-linear generalization of independent component analysis, replaces the standard assumption of independence for the joint distribution of coefficients with a distribution that is adapted to the variance structure of the coefficients of an efficient image basis. This offers a novel description of higher-order image structure and provides a way to learn coarse-coded, sparse-distributed representations of abstract image properties such as object location, scale, and texture.

## 1 Introduction

One of the major challenges in vision is how to derive from the retinal representation higher-order representations that describe properties of surfaces, objects, and scenes. Physiological studies of the visual system have characterized a wide range of response properties, beginning with, for example, simple cells and complex cells. These, however, offer only limited insight into how higher-order properties of images might be represented or even what the higher-order properties might be. Computational approaches to vision often derive algorithms by inverse graphics, i.e. by inverting models of the physics of light propagation and surface reflectance properties to recover object and scene properties. A drawback of this approach is that, because of the complexity of modeling, only the simplest and most approximate models are computationally feasible to invert and these often break down for realistic images. A more fundamental limitation, however, is that this formulation of the problem does not explain the adaptive nature of the visual system or how it can learn highly abstract and general representations of objects and surfaces.

An alternative approach is to derive representations from the statistics of the images themselves. This information theoretic view, called efficient coding, starts with the observation that there is an equivalence between the degree of structure represented and the efficiency of the code [1]. The hypothesis is that the primary goal of early sensory coding is to encode information efficiently. This theory has been applied to derive efficient codes for

---

[*]To whom correspondence should be addressed

natural images and to explain a wide range of response properties of neurons in the visual cortex [2–7].

Most algorithms for learning efficient representations assume either simply that the data are generated by a linear superposition of basis functions, as in independent component analysis (ICA), or, as in sparse coding, that the basis function coefficients are 'sparsified' by lateral inhibition. Clearly, these simple models are insufficient to capture the rich structure of natural images, and although they capture higher-order *statistics* of natural images (correlations beyond second order), it remains unclear how to go beyond this to discover higher-order image *structure*.

One approach is to learn image classes by embedding the statistical density assumed by ICA in a mixture model [8]. This provides a method for modeling classes of images and for performing automatic scene segmentation, but it assumes a fundamentally local representation and therefore is not suitable for compactly describing the large degree of structure variation across images. Another approach is to construct a specific model of non-linear features, e.g. the responses of complex cells, and learn an efficient code of their outputs [9]. With this, one is limited by the choice of the non-linearity and the range of image regularities that can be modeled.

In this paper, we take as a starting point the observation by Schwartz and Simoncelli [10] that, for natural images, there are significant statistical dependencies among the variances of filter outputs. By factoring out these dependencies with divisive normalization, Schwartz and Simoncelli showed that the model could account for a wide range of non-linearities observed in neurons in the auditory nerve and primary visual cortex.

Here, we propose a statistical model for higher-order structure that learns a basis on the variance regularities in natural images. This higher-order, non-orthogonal basis describes how, for a particular visual image patch, image basis function coefficient variances deviate from the default assumption of independence. This view offers a novel description of higher-order image structure and provides a way to learn sparse distributed representations of abstract image properties such as object location, scale, and surface texture.

**Efficient coding of natural images**

The computational goal of efficient coding is to derive from the statistics of the pattern ensemble a compact code that maximally reduces the redundancy in the patterns with minimal loss of information. The standard model assumes that the data is generated using a set of basis functions $\mathbf{A}$ and coefficients $\mathbf{u}$:

$$\mathbf{x} = \mathbf{A}\mathbf{u}, \tag{1}$$

Because coding efficiency is being optimized, it is necessary, either implicitly or explicitly, for the model to capture the probability distribution of the pattern ensemble. For the linear model, the data likelihood is [11, 12]

$$p(\mathbf{x}|\mathbf{A}) = p(\mathbf{u})/|\det \mathbf{A}|. \tag{2}$$

The coefficients $u_i$, are assumed to be statistically independent

$$p(\mathbf{u}) = \prod_i p(u_i). \tag{3}$$

ICA learns efficient codes of natural scenes by adapting the basis vectors to maximize the likelihood of the ensemble of image patterns, $p(\mathbf{x}_1, \ldots, \mathbf{x}_N) = \prod_n p(\mathbf{x}_n|\mathbf{A})$, which maximizes the independence of the coefficients and optimizes coding efficiency *within the limits of the linear model*.

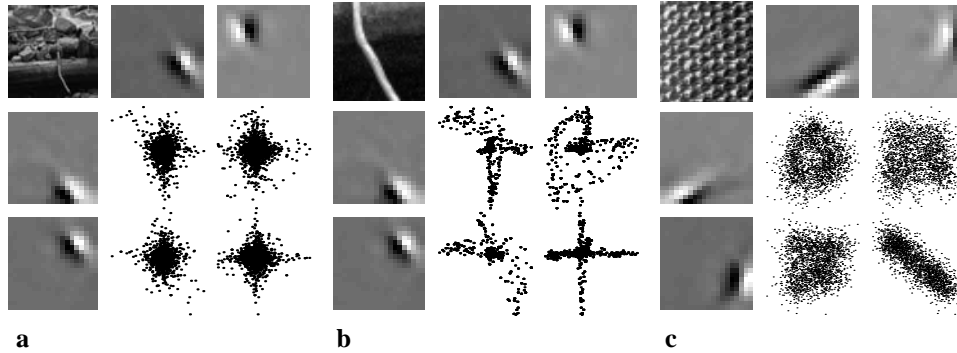

Figure 1: Statistical dependencies among natural image independent component basis co-efficients. The scatter plots show for the two basis functions in the same row and column the joint distributions of basis function coefficients. Each point represents the encoding of a $20 \times 20$ image patch centered at random locations in the image. (**a**) For complex natural scenes, the joint distributions appear to be independent, because the joint distribution can be approximated by the product of the marginals. (**b**) Closer inspection of particular image regions (the image in (b) is contained in the lower middle part of the image in (a)) reveals complex statistical dependencies for the same set of basis functions. (**c**) Images such as texture can also show complex statistical dependencies.

### Statistical dependencies among 'independent' components

A linear model can only achieve limited statistical independence among the basis function coefficients and thus can only capture a limited degree of visual structure. Deviations from independence among the coefficients reflect particular kinds of visual structure (fig. 1). If the coefficients were independent it would be possible to describe the joint distribution as the product of two marginal densities, $p(u_i, u_j) = p(u_i)p(u_j)$. This is approximately true for natural scenes (fig.1a), but for particular images, the joint distribution of coefficients show complex statistical dependencies that reflect the higher-order structure (figs.1b and 1c). The challenge for developing more general models of efficient coding is formulating a description of these higher-order correlations in a way that captures meaningful higher-order visual structure.

## 2 Modeling higher-order statistical structure

The basic model of standard efficient coding methods has two major limitations. First, the transformation from the pattern to the coefficients is linear, so only a limited class of computations can be achieved. Second, the model can capture statistical relationships among the pixels, but does not provide any means to capture higher order relationships that cannot be simply described at the pixel level. As a first step toward overcoming these limitations, we extend the basic model by introducing a non-independent prior to model higher-order statistical relationships among the basis function coefficients.

Given a representation of natural images in terms of a Gabor-wavelet-like representation learned by ICA, one salient statistical regularity is the covariation of basis function coefficients in different visual contexts. Any specific type of image region, e.g. a particular kind of texture, will tend to yield in large values for some coefficients and not others. Different types of image regions will exhibit different statistical regularities among the variances of the coefficients. For a large ensemble of images, the goal is to find a code that describes these higher-order correlations efficiently.

In the standard efficient coding model, the coefficients are often assumed to follow a generalized Gaussian distribution

$$p(u_i) = z e^{-|u_i/\lambda_i|^q}, \tag{4}$$

where $z = q/(2\lambda_i \Gamma[1/q])$. The exponent $q$ determines the distribution's shape and weight of the tails, and can be fixed or estimated from the data for each basis function coefficient. The parameter $\lambda_i$ determines the scale of variation (usually fixed in linear models, since basis vectors in $\mathbf{A}$ can absorb the scaling). $\lambda_i$ is a generalized notion of variance; for clarity, we refer to it simply as *variance* below.

Because we want to capture regularities among the variance patterns of the coefficients, we do not want to model the values of $\mathbf{u}$ themselves. Instead, we assume that the relative variances in different visual contexts can be modeled with a linear basis as follows

$$\lambda_i \quad = \quad \exp([\mathbf{Bv}]_i) \tag{5}$$
$$\Rightarrow \log \boldsymbol{\lambda} \quad = \quad \mathbf{Bv}. \tag{6}$$

where $[\mathbf{Bv}]_i$ refers to the $i^{th}$ element of the product vector $\mathbf{Bv}$. This formulation is useful because it uses a basis to represent the *deviation* from the variance assumed by the standard model. If we assume that $v_i$ also follows a zero-centered, sparse distribution (e.g. a generalized Gaussian), then $\mathbf{Bv}$ is peaked around zero which yields a variance of one, as in standard ICA. Because the distribution is sparse, only a few of the basis vectors in $\mathbf{B}$ are needed to describe how any particular image deviates from the default assumption of independence. The joint distribution for the prior (eqn.3) becomes

$$-\log p(\mathbf{u}|\mathbf{B},\mathbf{v}) \propto \sum_i^L \left| \frac{u_i}{e^{[\mathbf{Bv}]_i}} \right|^q, \tag{7}$$

Having formulated the problem as a statistical model, the choice of the value of $\mathbf{v}$ for a given $\mathbf{u}$ is determined by maximizing the posterior distribution

$$\hat{\mathbf{v}} = \arg\max_{\mathbf{v}} p(\mathbf{v}|\mathbf{u},\mathbf{B}) = \arg\max_{\mathbf{v}} p(\mathbf{u}|\mathbf{B},\mathbf{v}) p(\mathbf{v}) \tag{8}$$

Unfortunately, computing the most probable $\mathbf{v}$ is not straightforward. Because $\mathbf{v}$ specifies the variance of $\mathbf{u}$, there is a range of values that could account for a given pattern – all that changes is the probability of the first order representation, $p(\mathbf{u}|\mathbf{B},\mathbf{v})$. For the simulations below, $\hat{\mathbf{v}}$ was estimated by gradient ascent.

By maximizing the posterior $p(\mathbf{v}|\mathbf{u},\mathbf{B})$, the algorithm is computing the best way to describe how the distribution of $v_i$'s for the current image patch deviates from the default assumption of independence, i.e. $\mathbf{v} = 0$. This aspect of the algorithm makes the transformation from the data to the internal representation fundamentally non-linear. The basis functions in $\mathbf{B}$ represent an efficient, sparse, distributed code for commonly observed deviations. In contrast to the first layer, where basis functions in $\mathbf{A}$ correspond to specific visual features, higher-order basis functions in $\mathbf{B}$ describe the shapes of image *distributions*.

The parameters are adapted by performing gradient ascent on the data likelihood. Using the generalized prior, the data likelihood is computed by marginalizing over the coefficients. Assuming independence between $\mathbf{B}$ and $\mathbf{v}$, the marginal likelihood is

$$p(\mathbf{x}|\mathbf{A},\mathbf{B}) = \int p(\mathbf{u}|\mathbf{B},\mathbf{v}) p(\mathbf{v})/|\det \mathbf{A}| d\mathbf{v}. \tag{9}$$

This, however, is intractable to compute, so we approximate it by the maximum a posteriori value $\hat{\mathbf{v}}$

$$p(\mathbf{x}|\mathbf{A},\mathbf{B}) \approx p(\mathbf{u}|\mathbf{B},\hat{\mathbf{v}}) p(\hat{\mathbf{v}})/|\det \mathbf{A}|. \tag{10}$$

We assume that $p(\mathbf{v}) = \prod_i p(v_i)$ and that $p(v_i) \propto \exp(-|v_i|)$. We adapt $\mathbf{B}$ by maximizing the likelihood over the data ensemble

$$\mathbf{B} = \arg\max_{\mathbf{B}} \sum_n \log p(\mathbf{u}_n|\mathbf{B},\hat{\mathbf{v}}_n) + \log p(\mathbf{B}) \tag{11}$$

For reasons of space, we omit the (straightforward) derivations of the gradients.

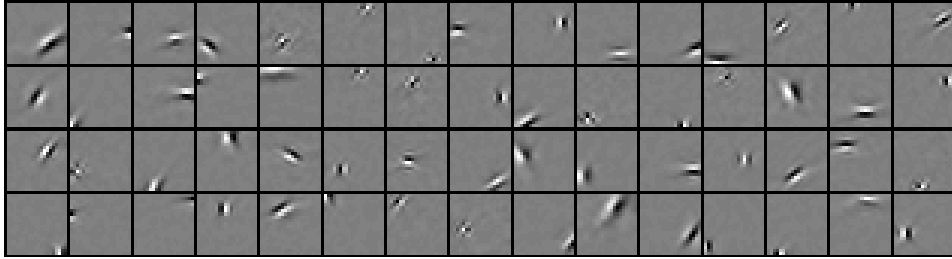

Figure 2: A subset of the 400 image basis functions. Each basis function is 20x20 pixels.

## 3   Results

The algorithm described above was applied to a standard set of ten $512 \times 512$ natural images used in [2]. For computational simplicity, prior to the adaptation of the higher-order basis **B**, a $20 \times 20$ ICA image basis was derived using standard methods (e.g. [3]). A subset of these basis functions is shown in fig. 2.

Because of the computational complexity of the learning procedure, the number of basis functions in **B** was limited to 30, although in principle a complete basis of 400 could be learned. The basis **B** was initialized to small random values and gradient ascent was performed for 4000 iterations, with a fixed step size of 0.05. For each batch of 5000 randomly sampled image patches, $\hat{\mathbf{v}}$ was derived using 50 steps of gradient ascent at a fixed step size of 0.01.

Fig. 3 shows three different representations of the basis functions in the matrix **B** adapted to natural images. The first $10 \times 3$ block (fig.3a) shows the values of the 30 basis functions in **B** in their original learned order. Each square represents 400 weights $\mathbf{B}_{i,j}$ from a particular $v_j$ to all the image basis functions $u_i$'s. Black dots represent negative weights; white, positive weights. In this representation, the weights appear sparse, but otherwise show no apparent structure, simply because basis functions in **A** are unordered.

Figs. 3b and 3c show the weights rearranged in two different ways. In fig. 3b, the dots representing the same weights are arranged according to the spatial location within an image patch (as determined by fitting a 2D Gabor function) of the basis function which the weight affects. Each weight is shown as a dot; white dots represent positive weights, black dots negative weights. In fig. 3c, the same weights are arranged according to the orientation and spatial scale of the Gaussian envelope of the fitted Gabor. Orientation ranges from 0 to $\pi$ counter-clockwise from the horizontal axis, and spatial scale ranges radially from DC at the bottom center to Nyquist. (Note that the learned basis functions can only be approximately fit by Gabor functions, which limits the precision of the visualizations.)

In these arrangements, several types of higher-order regularities emerge. The predominant one is that coefficient variances are spatially correlated, which reflects the fact that a common occurrence is an image patch with a small localized object against a relatively uniform background. For example, the pattern in row 5, column 3 of fig. 3b shows that often the coefficient variances in the top and bottom halves of the image patch are anti-correlated, i.e. either the object or scene is primarily across the top or across the bottom. Because $v_i$ can be positive or negative, the higher-order basis functions in **B** represent contrast in the variance patterns. Other common regularities are variance-contrasts between two orientations for all spatial positions (e.g. row 7, column 1) and between low and high spatial scales for all positions and orientations (e.g. row 9, column 3). Most higher-order basis functions have simple structure in either position, orientation, or scale, but there are some whose organization is less obvious.

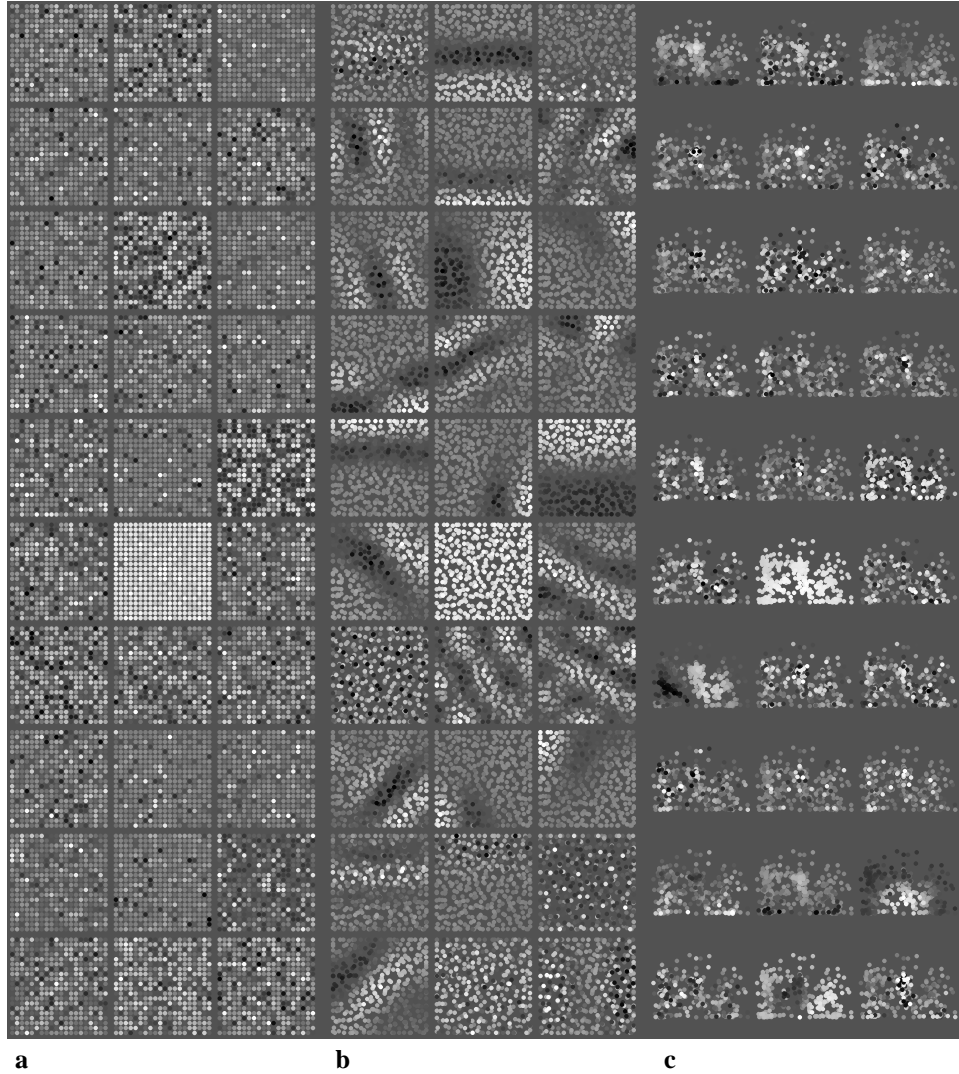

a                              b                              c

Figure 3: The learned higher-order basis functions. The same weights shown in the original order (a); rearranged according to the spatial location of the corresponding image basis functions (b); rearranged according to frequency and orientation of image basis functions (c). See text for details.

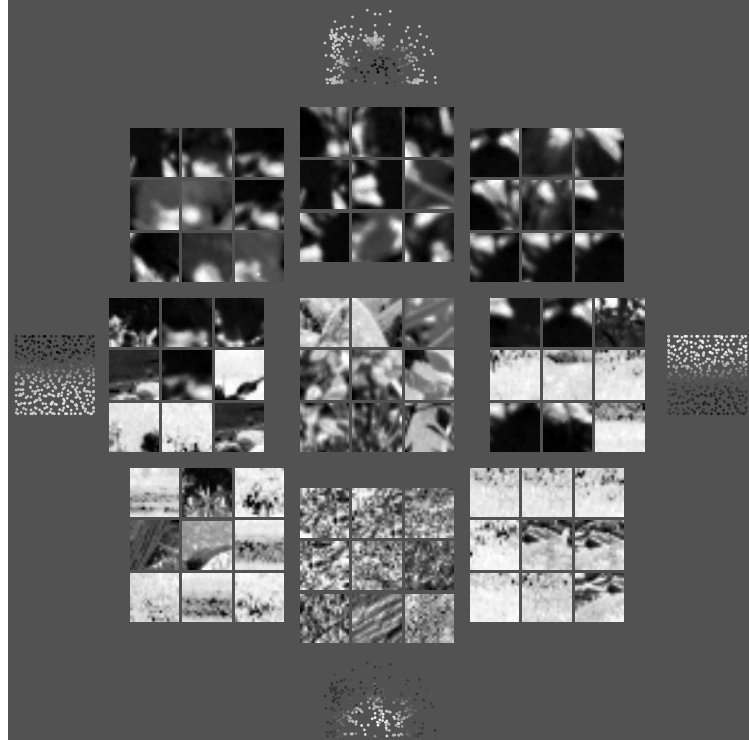

Figure 4: Image patches that yielded the largest coefficients for two basis functions in **B**. The central block contains nine image patches corresponding to higher-order basis function coefficients with values near zero, i.e. small deviations from independent variance patterns. Positions of other nine-patch blocks correspond to the associated values of higher-order coefficients, here $v_{15}$ and $v_{27}$ (whose weights to $u_i$'s are shown at the axes extrema). For example, the upper-left block contains image patches for which $v_{15}$ was highly negative (contrast localized to bottom half of patch) and $v_{27}$ was highly positive (power predominantly at low spatial scales). This illustrates how different combinations of basis functions in **B** define *distributions* of images (in this case, spatial frequency and location).

Another way to get insight into the code learned by the model is to display, for a large ensemble of image patches, the patches that yield the largest values of particular $v_i$'s (and their corresponding basis functions in **B**). This is shown in fig. 4.

As a check to see if any of the higher-order structure learned by the algorithm was simply due to random variations in the dataset, we generated a dataset by drawing independent samples $\mathbf{u}_n$ from a generalized Gaussian to produce the pattern $\mathbf{x}_n = \mathbf{A}\mathbf{u}_n$. The resulting basis **B** was composed only of small random values, indicating essentially no deviation from the standard assumption of independence and unit variance. In addition, adapting the model on a synthetic dataset generated from a hand-specified **B** recovers the original higher-order basis functions.

It is also possible to adapt **A** and **B** simultaneously (although with considerably greater computational expense). To check the validity of first deriving **B** for a fixed **A**, both matrices were adapted simultaneously for small $8 \times 8$ patches on the same natural image data set. The results for both the image basis matrix **A** and the higher-order basis **B** were qualitatively similar to those reported above.

# 4 Discussion

We have presented a model for learning higher-order statistical regularities in natural images by learning an efficient, sparse-distributed code for the basis function coefficient variances. The recognition algorithm is non-linear, but we have not tested yet whether it can account for non-linearities similar to the types reported in [10].

A (cautious) neurobiological interpretation of the higher-order units is that they are analogous to complex cells which pool output over specific first-order feature dimensions. Rather than achieving a simplistic invariance, however, the model presented here has the specific goal of efficiently representing the higher-order structure by adapting to the statistics of natural images, and thus may predict a broader range of response properties than are commonly tested physiologically.

One salient type of higher-order structure learned by the model is the position of image structure within the patch. It is interesting that, rather than encoding specific locations, the model learned a coarse code of position using broadly tuned spatial patterns. This could offer novel insights into the function of the broad tuning of higher level visual neurons. By learning higher-order basis functions for different classes of visual images, the model could not only provide insights into other types of visual response properties, but could provide a way to simplify some of the computations in perceptual organization and other computations in mid-level vision.

## References

[1] H. B. Barlow. Possible principles underlying the transformation of sensory messages. In W. A. Rosenbluth, editor, *Sensory Communication*, pages 217–234. MIT Press, Cambridge, 1961.

[2] B. A. Olshausen and D. J. Field. Emergence of simple-cell receptive-field properties by learning a sparse code for natural images. *Nature*, 381:607–609, 1996.

[3] A. J. Bell and T. J. Sejnowski. The 'independent components' of natural scenes are edge filters. *Vision Res.*, 37(23):3327–3338, 1997.

[4] J. H. van Hateren and A. van der Schaaf. Independent component filters of natural images compared with simple cells in primary visual cortex. *Proc. Royal Soc. Lond.* **B**, 265:359–366, 1998.

[5] J. H. van Hateren and D. L. Ruderman. Independent component analysis of natural image sequences yield spatiotemporal filters similar to simple cells in primary visual cortex. *Proc. Royal Soc. Lond. B*, 265:2315–2320, 1998.

[6] P. O. Hoyer and A. Hyvarinen. Independent component analysis applied to feature extraction from colour and stereo images. *Network*, 11(3):191–210, 2000.

[7] E. Simoncelli and B. Olshausen. Natural image statistics and neural representation. *Ann. Rev. Neurosci.*, 24:1193–1216, 2001.

[8] T-W. Lee and M. S. Lewicki. Unsupervised classification, segmentation and de-noising of images using ICA mixture models. *IEEE Trans. Image Proc.*, 11(3):270–279, 2002.

[9] P. O. Hoyer and A. Hyvarinen. A multi-layer sparse coding network learns contour coding from natural images. *Vision Research*, 42(12):1593–1605, 2002.

[10] O. Schwartz and E. P. Simoncelli. Natural signal statistics and sensory gain control. *Nat. Neurosci.*, 4:819–825, 2001.

[11] B. A. Pearlmutter and L. C. Parra. A context-sensitive generalization of ICA. In *International Conference on Neural Information Processing*, pages 151–157, 1996.

[12] J-F. Cardoso. Infomax and maximum likelihood for blind source separation. *IEEE Signal Processing Letters*, 4:109–111, 1997.
